# Variational Inference for Diffusion Processes

**Cédric Archambeau**
University College London
c.archambeau@cs.ucl.ac.uk

**Manfred Opper**
Technical University Berlin
opperm@cs.tu-berlin.de

**Yuan Shen**
Aston University
y.shen2@aston.ac.uk

**Dan Cornford**
Aston University
d.cornford@aston.ac.uk

**John Shawe-Taylor**
University College London
jst@cs.ucl.ac.uk

## Abstract

Diffusion processes are a family of continuous-time continuous-state stochastic processes that are in general only partially observed. The joint estimation of the forcing parameters and the system noise (volatility) in these dynamical systems is a crucial, but non-trivial task, especially when the system is nonlinear and multi-modal. We propose a variational treatment of diffusion processes, which allows us to compute type II maximum likelihood estimates of the parameters by simple gradient techniques and which is computationally less demanding than most MCMC approaches. We also show how a cheap estimate of the posterior over the parameters can be constructed based on the variational free energy.

## 1 Introduction

Continuous-time diffusion processes, described by stochastic differential equations (SDEs), arise naturally in a range of applications from environmental modelling to mathematical finance [13]. In statistics the problem of Bayesian inference for both the state and parameters, within partially observed, non-linear diffusion processes has been tackled using Markov Chain Monte Carlo (MCMC) approaches based on data augmentation [17, 11], Monte Carlo exact simulation methods [6], or Langevin / hybrid Monte Carlo methods [1, 3]. Within the signal processing community solutions to the so called Zakai equation [12] based on particle filters [8], a variety of extensions to the Kalman filter/smoother [2, 5] and mean field analysis of the SDE together with moment closure methods [10] have also been proposed. In this work we develop a novel variational approach to the problem of approximate inference in continuous-time diffusion processes, including a marginal likelihood (evidence) based inference technique for the forcing parameters. In general, joint parameter and state inference using naive methods is complicated due to dependencies between state and system noise parameters.

We work in continuous time, computing *distributions over sample paths*[1], and discretise only in our posterior approximation, which has advantages over methods based on discretising the SDE directly [3]. The approximate inference approach we describe is more computationally efficient than competing Monte Carlo algorithms and could be further improved in speed by defining a variety of sub-optimal approximations. The approximation is also more accurate than existing Kalman smoothing methods applied to non-linear systems [4]. Ultimately, we are motivated by the critical requirement to estimate parameters within large environmental models, where at present only a small number of Kalman filter/smoother based estimation algorithms have been attempted [2], and there have been no likelihood based attempts to estimate the system noise forcing parameters.

In Section 2 and 3, we introduce the formalism for a variational treatment of partially observed diffusion processes with measurement noise and we provide the tools to estimate the optimal variational posterior process [4]. Section 4 deals with the estimation of the drift and the system noise parameter, as well as the estimation of the optimal initial conditions. Finally, the approach is validated on a bi-stable nonlinear system in Section 5. In this context, we also discuss how to construct an estimate of the posterior distribution over parameters based on the variational free energy.

## 2    Diffusion processes with measurement error

Consider the continuous-time continuous-state stochastic process $X = \{X_t, \ t_0 \leq t \leq t_f\}$. We assume this process is a $d$-dimensional diffusion process. Its time evolution is described by the following SDE (to be interpreted as an Ito stochastic integral):

$$dX_t = \mathbf{f}_{\boldsymbol{\theta}}(t, X_t) \ dt + \boldsymbol{\Sigma}^{1/2} \ dW_t, \quad dW_t \sim \mathcal{N}(\mathbf{0}, dt\mathbf{I}). \tag{1}$$

The nonlinear vector function $\mathbf{f}_{\boldsymbol{\theta}}$ defines the deterministic drift and the positive semi-definite matrix $\boldsymbol{\Sigma} \in \mathbb{R}^{d \times d}$ is the system noise covariance. The diffusion is modelled by a $d$-dimensional Wiener process $W = \{W_t, \ t_0 \leq t \leq t_f\}$ (see e.g. [13] for a formal definition). Eq. (1) defines a process with additive system noise. This might seem restrictive at first sight. However, it can be shown [13, 17, 6] that a range of state dependent stochastic forcings can be transformed into this form.

It is further assumed that only a small number of discrete-time latent states are observed and that the observations are subject to measurement error. We denote the set of observations at the discrete times $\{t_n\}_{n=1}^N$ by $Y = \{\mathbf{y}_n\}_{n=1}^N$ and the corresponding latent states by $\{\mathbf{x}_n\}_{n=1}^N$, with $\mathbf{x}_n = X_{t=t_n}$. For simplicity, the measurement noise is modelled by a zero-mean multivariate Gaussian density, with covariance matrix $\mathbf{R} \in \mathbb{R}^{d \times d}$.

## 3    Approximate inference for diffusion processes

Our approximate inference scheme builds on [4] and is based on a variational inference approach (see for example [7]). The aim is to minimise the variational free energy, which is defined as follows:

$$\mathcal{F}_{\boldsymbol{\Sigma}}(q, \boldsymbol{\theta}) = - \left\langle \ln \frac{p(Y, X | \boldsymbol{\theta}, \boldsymbol{\Sigma})}{q(X | \boldsymbol{\Sigma})} \right\rangle_q, \quad X = \{X_t, \ t_0 \leq t \leq t_f\}, \tag{2}$$

where $q(X | \boldsymbol{\Sigma})$ is an approximate posterior process *over sample paths* in the interval $[t_0, t_f]$ and $\boldsymbol{\theta}$ are the parameters, excluding the stochastic forcing covariance matrix $\boldsymbol{\Sigma}$. Hence, this quantity is an upper bound to the negative log-marginal likelihood:

$$-\ln p(Y | \boldsymbol{\theta}, \boldsymbol{\Sigma}) = \mathcal{F}_{\boldsymbol{\Sigma}}(q, \boldsymbol{\theta}) - \mathrm{KL}\left[q(X | \boldsymbol{\Sigma}) \| p(X | Y, \boldsymbol{\theta}, \boldsymbol{\Sigma})\right] \leq \mathcal{F}_{\boldsymbol{\Sigma}}(q, \boldsymbol{\theta}). \tag{3}$$

As noted in Appendix A, this bound is finite if the approximate process is another diffusion process with a system noise covariance chosen to be identical to that of the prior process induced by (1).

The standard approach for learning the parameters in presence of latent variables is to use an EM type algorithm [9]. However, since the variational distribution is restricted to have the same system noise covariance (see Appendix A) as the true posterior, the EM algorithm would leave this covariance completely unchanged in the M step and cannot be used for learning this crucial parameter. Therefore, we adopt a different approach, which is based on a conjugate gradient method.

### 3.1    Optimal approximate posterior process

We consider an approximate time-varying linear process with the same diffusion term, that is the same system noise covariance:

$$dX_t = \mathbf{g}(t, X_t) \ dt + \boldsymbol{\Sigma}^{1/2} \ dW_t, \quad dW_t \sim \mathcal{N}(\mathbf{0}, dt\mathbf{I}), \tag{4}$$

where $\mathbf{g}(t, \mathbf{x}) = -\mathbf{A}(t)\mathbf{x} + \mathbf{b}(t)$, with $\mathbf{A}(t) \in \mathbb{R}^{d \times d}$ and $\mathbf{b}(t) \in \mathbb{R}^d$. In other words, the approximate posterior process $q(X | \boldsymbol{\Sigma})$ is restricted to be a Gaussian process [4]. The Gaussian marginal at time $t$ is defined as follows:

$$q(X_t | \boldsymbol{\Sigma}) = \mathcal{N}(X_t | \mathbf{m}(t), \mathbf{S}(t)), \quad t_0 \leq t \leq t_f, \tag{5}$$

where $\mathbf{m}(t) \in \mathbb{R}^d$ and $\mathbf{S}(t) \in \mathbb{R}^{d \times d}$ are respectively the marginal mean and the marginal covariance at time $t$. In the rest of the paper, we denote $q(X_t|\mathbf{\Sigma})$ by the shorthand notation $q_t$.

For fixed parameters $\boldsymbol{\theta}$ and assuming that there is no observation at the initial time $t_0$, the optimal approximate posterior process $q(X|\mathbf{\Sigma})$ is the one minimizing the variational free energy, which is given by (see Appendix A)

$$\mathcal{F}_{\mathbf{\Sigma}}(q, \boldsymbol{\theta}) = \int_{t_0}^{t_f} E_{sde}(t) \, dt + \int_{t_0}^{t_f} E_{obs}(t) \sum_n \delta(t - t_n) \, dt + \text{KL}\left[q_0 \| p_0\right]. \tag{6}$$

The function $\delta(t)$ is Dirac's delta function. The energy functions $E_{sde}(t)$ and $E_{obs}(t)$ are defined as follows:

$$E_{sde}(t) = \frac{1}{2} \left\langle (\mathbf{f}_{\boldsymbol{\theta}}(t, X_t) - \mathbf{g}(t, X_t))^\top \mathbf{\Sigma}^{-1} (\mathbf{f}_{\boldsymbol{\theta}}(t, X_t) - \mathbf{g}(t, X_t)) \right\rangle_{q_t}, \tag{7}$$

$$E_{obs}(t) = \frac{1}{2} \left\langle (Y_t - X_t)^\top \mathbf{R}^{-1} (Y_t - X_t) \right\rangle_{q_t} + \frac{d}{2} \ln 2\pi + \frac{1}{2} \ln |\mathbf{R}|. \tag{8}$$

where $\{Y_t, t_0 \le t \le t_f\}$ is the underlying continuous-time observable process.

## 3.2 Smoothing algorithm

The variational parameters to optimise in order to find the optimal Gaussian process approximation are $\mathbf{A}(t)$, $\mathbf{b}(t)$, $\mathbf{m}(t)$ and $\mathbf{S}(t)$. For a linear SDE with additive system noise, it can be shown that the time evolution of the means and the covariances are described by a set of ordinary differential equations [13, 4]:

$$\dot{\mathbf{m}}(t) = -\mathbf{A}(t)\mathbf{m}(t) + \mathbf{b}(t), \tag{9}$$

$$\dot{\mathbf{S}}(t) = -\mathbf{A}(t)\mathbf{S}(t) - \mathbf{S}(t)\mathbf{A}^\top(t) + \mathbf{\Sigma}, \tag{10}$$

where $\dot{}$ denotes the time derivtive. These equations provide us with consistency constraints for the marginal means and the marginal covariances along sample paths. To enforce these constraints we formulate the Lagrangian

$$\mathcal{L}_{\boldsymbol{\theta}, \mathbf{\Sigma}} = \mathcal{F}_{\mathbf{\Sigma}}(q, \boldsymbol{\theta}) - \int_{t_0}^{t_f} \boldsymbol{\lambda}^\top(t) \left( \dot{\mathbf{m}}(t) + \mathbf{A}(t)\mathbf{m}(t) - \mathbf{b}(t) \right) dt$$

$$- \int_{t_0}^{t_f} \text{tr} \left\{ \mathbf{\Psi}(t) \left( \dot{\mathbf{S}}(t) + 2\mathbf{A}(t)\mathbf{S}(t) - \mathbf{\Sigma} \right) \right\} dt, \tag{11}$$

where $\boldsymbol{\lambda}(t) \in \mathbb{R}^d$ and $\mathbf{\Psi}(t) \in \mathbb{R}^{d \times d}$ are time dependent Lagrange multipliers, with $\mathbf{\Psi}(t)$ symmetric.

First, taking the functional derivatives of $\mathcal{L}_{\boldsymbol{\theta}, \mathbf{\Sigma}}$ with respect to $\mathbf{A}(t)$ and $\mathbf{b}(t)$ results in the following gradient functions:

$$\nabla_{\mathbf{A}} \mathcal{L}_{\boldsymbol{\theta}, \mathbf{\Sigma}}(t) = \nabla_{\mathbf{A}} E_{sde}(t) - \boldsymbol{\lambda}(t)\mathbf{m}^\top(t) - 2\mathbf{\Psi}(t)\mathbf{S}(t), \tag{12}$$

$$\nabla_{\mathbf{b}} \mathcal{L}_{\boldsymbol{\theta}, \mathbf{\Sigma}}(t) = \nabla_{\mathbf{b}} E_{sde}(t) + \boldsymbol{\lambda}(t). \tag{13}$$

The gradients $\nabla_{\mathbf{A}} E_{sde}(t)$ and $\nabla_{\mathbf{b}} E_{sde}(t)$ are derived in Appendix B.

Secondly, taking the functional derivatives of $\mathcal{L}_{\boldsymbol{\theta}, \mathbf{\Sigma}}$ with respect to $\mathbf{m}(t)$ and $\mathbf{S}(t)$, setting to zero and rearranging leads to a set of ordinary differential equations, which describe the time evolution of the Lagrange multipliers, along with jump conditions when there are observations:

$$\dot{\boldsymbol{\lambda}}(t) = -\nabla_{\mathbf{m}} E_{sde}(t) + \mathbf{A}^\top(t)\boldsymbol{\lambda}(t), \quad \boldsymbol{\lambda}_n^+ = \boldsymbol{\lambda}_n^- - \nabla_{\mathbf{m}} E_{obs}(t)|_{t=t_n}, \tag{14}$$

$$\dot{\mathbf{\Psi}}(t) = -\nabla_{\mathbf{S}} E_{sde}(t) + 2\mathbf{\Psi}(t)\mathbf{A}(t), \quad \mathbf{\Psi}_n^+ = \mathbf{\Psi}_n^- - \nabla_{\mathbf{S}} E_{obs}(t)|_{t=t_n}. \tag{15}$$

The optimal variational functions can be computed by means of a gradient descent technique, such as the conjugate gradient (see e.g., [16]). The explicit gradients with respect to $\mathbf{A}(t)$ and $\mathbf{b}(t)$ are given by (12) and (13). Since $\mathbf{m}(t)$, $\mathbf{S}(t)$, $\boldsymbol{\lambda}(t)$ and $\mathbf{\Psi}(t)$ are dependent on these parameters, one needs also to take the corresponding implicit derivatives into account. However, these implicit gradients vanish if the consistency constraints for the means (9) and the covariances (10), as well as the ones for the Lagrange multipliers (14-15), are satisfied. One way to achieve this is to perform a forward propagation of the means and the covariances, followed by a backward propagation of the Lagrange multipliers, and then to take a gradient step. The resulting algorithm for computing the optimal posterior $q(X|\mathbf{\Sigma})$ over sample paths is detailed in Algorithm 1.

**Algorithm 1** Compute the optimal $q(X|\boldsymbol{\Sigma})$.

1: $\textbf{input}(\mathbf{m}_0, \mathbf{S}_0, \boldsymbol{\theta}, \boldsymbol{\Sigma}, t_0, t_f, \Delta t, \omega)$
2: $K \leftarrow (t_f - t_0)/\Delta t$
3: initialise $\{\mathbf{A}_k, \mathbf{b}_k\}_{k \geq 0}$
4: **repeat**
5:     **for** $k = 0$ to $K - 1$ **do**
6:         $\mathbf{m}_{k+1} \leftarrow \mathbf{m}_k - (\mathbf{A}_k \mathbf{m}_k - \mathbf{b}_k)\Delta t$
7:         $\mathbf{S}_{k+1} \leftarrow \mathbf{S}_k - (\mathbf{A}_k \mathbf{S}_k + \mathbf{S}_k \mathbf{A}_k^\top - \boldsymbol{\Sigma})\Delta t$
8:     **end for**{forward propagation}
9:     **for** $k = K$ to $1$ **do**
10:         $\boldsymbol{\lambda}_{k-1} \leftarrow \boldsymbol{\lambda}_k + (\nabla_{\mathbf{m}} E_{sde}|_{t=t_k} - \mathbf{A}_k^\top \boldsymbol{\lambda}_k)\Delta t$
11:         $\boldsymbol{\Psi}_{k-1} \leftarrow \boldsymbol{\Psi}_k + (\nabla_{\mathbf{S}} E_{sde}|_{t=t_k} - 2\boldsymbol{\Psi}_k \mathbf{A}_k)\Delta t$
12:         **if** observation at $t_{k-1}$ **then**
13:             $\boldsymbol{\lambda}_{k-1} \leftarrow \boldsymbol{\lambda}_{k-1} + \nabla_{\mathbf{m}} E_{obs}|_{t=t_{k-1}}$
14:             $\boldsymbol{\Psi}_{k-1} \leftarrow \boldsymbol{\Psi}_{k-1} + \nabla_{\mathbf{S}} E_{obs}|_{t=t_{k-1}}$
15:         **end if**{jumps}
16:     **end for**{backward sweep (adjoint operation)}
17:     update $\{\mathbf{A}_k, \mathbf{b}_k\}_{k \geq 0}$ using the gradient functions (12) and (13)
18: **until** minimum of $\mathcal{L}_{\boldsymbol{\theta}, \boldsymbol{\Sigma}}$ is attained {optimisation loop}
19: **return** $\{\mathbf{A}_k, \mathbf{b}_k, \mathbf{m}_k, \mathbf{S}_k, \boldsymbol{\lambda}_k, \boldsymbol{\Psi}_k\}_{k \geq 0}$

## 4 Parameter estimation

The parameters to optimise include the parameters of the prior over the initial state, the drift function parameters and the system noise covariance. The estimation of the parameters related to the observable process are not discussed in this work, although it is a straightforward extension.

The smoothing algorithm described in the previous section computes the optimal posterior process by providing us with the stationary solution functions $\mathbf{A}(t)$ and $\mathbf{b}(t)$. Therefore, when subsequently optimising the parameters we only need to compute their explicit derivatives; the implicit ones vanish since $\nabla_{\mathbf{A}} \mathcal{L}_{\boldsymbol{\theta}, \boldsymbol{\Sigma}} = 0$ and $\nabla_{\mathbf{b}} \mathcal{L}_{\boldsymbol{\theta}, \boldsymbol{\Sigma}} = 0$. Before computing the gradients, we integrate (11) by parts to make the boundary conditions explicit. This leads to

$$\mathcal{L}_{\boldsymbol{\theta}, \boldsymbol{\Sigma}} = \mathcal{F}_{\boldsymbol{\Sigma}}(q, \boldsymbol{\theta}) - \int_{t_0}^{t_f} \left\{ \boldsymbol{\lambda}^\top(t)\big(\mathbf{A}(t)\mathbf{m}(t) - \mathbf{b}(t)\big) - \dot{\boldsymbol{\lambda}}^\top(t)\mathbf{m}(t) \right\} dt - \boldsymbol{\lambda}_f^\top \mathbf{m}_f + \boldsymbol{\lambda}_0^\top \mathbf{m}_0$$

$$- \int_{t_0}^{t_f} \text{tr} \left\{ \boldsymbol{\Psi}(t)\big(2\mathbf{A}(t)\mathbf{S}(t) - \boldsymbol{\Sigma}\big) - \dot{\boldsymbol{\Psi}}(t)\mathbf{S}(t) \right\} dt - \text{tr}\{\boldsymbol{\Psi}_f \mathbf{S}_f\} + \text{tr}\{\boldsymbol{\Psi}_0 \mathbf{S}_0\}, \quad (16)$$

At the final time $t_f$, there are no consistency constraints, that is $\boldsymbol{\lambda}_f$ and $\boldsymbol{\Psi}_f$ are both equal to zero.

### 4.1 Initial state

The initial variational posterior $q(\mathbf{x}_0)$ is chosen equal to $\mathcal{N}(\mathbf{x}_0|\mathbf{m}_0, \mathbf{S}_0)$ to ensure that the approximate process is a Gaussian one. Taking the derivatives of (16) with respect to $\mathbf{m}_0$ and $\mathbf{S}_0$ results in the following expressions:

$$\nabla_{\mathbf{m}_0} \mathcal{L}_{\boldsymbol{\theta}, \boldsymbol{\Sigma}} = \boldsymbol{\lambda}_0 + \tau_0^{-1}(\mathbf{m}_0 - \boldsymbol{\mu}_0), \quad \nabla_{\mathbf{S}_0} \mathcal{L}_{\boldsymbol{\theta}, \boldsymbol{\Sigma}} = \boldsymbol{\Psi}_0 + \frac{1}{2}\left(\tau_0^{-1}\mathbf{I} - \mathbf{S}_0^{-1}\right), \quad (17)$$

where the prior $p(\mathbf{x}_0)$ is assumed to be an isotropic Gaussian density with mean $\boldsymbol{\mu}_0$. Its variance $\tau_0$ is taken sufficiently large to give a broad prior.

### 4.2 Drift

The gradients for the drift function parameters $\boldsymbol{\theta}_f$ only depend on the total energy associated to the SDE. Their general expression is given by

$$\nabla_{\boldsymbol{\theta}_f} \mathcal{L}_{\boldsymbol{\theta}, \boldsymbol{\Sigma}} = \int_{t_0}^{t_f} \nabla_{\boldsymbol{\theta}_f} E_{sde}(t) \, dt, \quad (18)$$

where $\nabla_{\boldsymbol{\theta}_f} E_{sde}(t) = \left\langle \left(\mathbf{f}_{\boldsymbol{\theta}}(t, X_t) - \mathbf{g}(t, X_t)\right)^\top \boldsymbol{\Sigma}^{-1} \nabla_{\boldsymbol{\theta}_f} \mathbf{f}_{\boldsymbol{\theta}}(t, X_t) \right\rangle_{q_t}$. Note that the observations do play a role in this gradient as they enter through $\mathbf{g}(t, X_t)$ and the expectation w.r.t. $q(X_t | \boldsymbol{\Sigma})$.

### 4.3 System noise

Estimating the system noise covariance (or volatility) is essential as the system noise, together with the drift function, determines the dynamics. In general, this parameter is difficult to estimate using an MCMC approach because the efficiency is strongly dependent on the discrete approximation of the SDE and most methods break down when the time step $\Delta t$ gets too small [11, 6]. For example in a Bayesian MCMC approach, which alternates between sampling paths and parameters, the latent paths imputed between observations must have a system noise parameter which is arbitrarily close to its previous value in order to be accepted by a Metropolis sampler. Hence, the algorithm becomes extremely slow. Note, that for the same reason, a naive EM algorithm within our approach breaks down. However, in our method, we can simply compute approximations to the marginal likelihood and its gradient directly. In the next section, we will compare our results to a direct MCMC estimate of the marginal likelihood which is a time consuming method.

The gradient of (16) with respect to $\boldsymbol{\Sigma}$ is given by

$$\nabla_{\boldsymbol{\Sigma}} \mathcal{L}_{\boldsymbol{\theta}, \boldsymbol{\Sigma}} = \int_{t_0}^{t_f} \nabla_{\boldsymbol{\Sigma}} E_{sde}(t) \, dt + \int_{t_0}^{t_f} \boldsymbol{\Psi}(t) \, dt, \tag{19}$$

where $\nabla_{\boldsymbol{\Sigma}} E_{sde}(t) = -\frac{1}{2} \boldsymbol{\Sigma}^{-1} \left\langle \left(\mathbf{f}_{\boldsymbol{\theta}}(t, X_t) - \mathbf{g}(t, X_t)\right)\left(\mathbf{f}_{\boldsymbol{\theta}}(t, X_t) - \mathbf{g}(t, X_t)\right)^\top \right\rangle_{q_t} \boldsymbol{\Sigma}^{-1}$.

## 5 Experimental validation on a bi-stable system

In order to validate the approach, we consider the 1 dimensional double-well system:

$$f_\theta(t, x) = 4x\left(\theta - x^2\right), \quad \theta > 0, \tag{20}$$

where $f_\theta(t, x)$ is the drift function. This dynamical system is highly nonlinear and its stationary distribution is multi-modal. It has two stable states, one in $x = -\theta$ and one in $x = +\theta$. The system is driven by the system noise, which makes it occasionally flip from one well to the other.

In the experiments, we set the drift parameter $\theta$ to 1, the system noise standard deviation $\sigma$ to 0.5 and the measurement error standard deviation $r$ to 0.2. The time step for the variational approximation is set to $\Delta t = 0.01$, which is identical to the time resolution used to generate the original sample path. In this setting, the exit time from one of the wells is 4000 time units [15]. In other words, the transition from one well to the other is highly unlikely in the window of roughly 8 time units that we consider and where a transition occurs.

Figure 1(a) compares the variational solution to the outcomes of a hybrid MCMC simulation of the posterior process using the true parameter values. The hybrid MCMC approach was proposed in [1]. At each step of the sampling process, an entire sample path is generated. In order to keep the acceptance of new paths sufficiently high, the basic MCMC algorithm is combined with ideas from Molecular Dynamics, such that the MCMC sampler moves towards regions of high probability in the state space. An important drawback of MCMC approaches is that it might be extremely difficult to monitor their convergence and that they may require a very large number of samples before actually converging. In particular, over $100,000$ sample paths were necessary to reach convergence in the case of the double-well system.

The solution provided by the hybrid MCMC is here considered as the base line solution. One can observe that the variational solution underestimates the uncertainty (smaller error bars). Nevertheless, the time of the transition is correctly located. Convergence of the smoothing algorithm was achieved in approximately 180 conjugate gradient steps, each one involving a forward and backward sweep.

The optimal parameters and the optimal initial conditions for the variational solution are given by

$$\hat{\sigma} = 0.72, \quad \hat{\theta}_f = 0.85, \quad \hat{m}_0 = 0.88, \quad \hat{s}_0 = 0.45. \tag{21}$$

Convergence of the outer optimization loop is typically reached after less then 10 conjugate gradient steps. While the estimated value for the drift parameter is within $15\%$ percent from its true value,

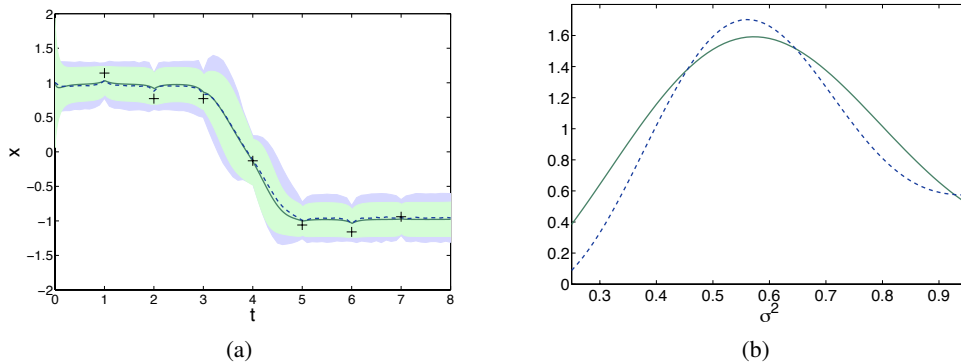

(a)

(b)

Figure 1: (a) Variational solution (solid) compared to the hybrid MCMC solution (dashed), using the true parameter values. The curves denote the mean paths and the shaded regions are the two-standard deviation noise tubes. (b) Posterior of the system noise variance (diffusion term). The plain curve and the dashed curve are respectively the approximations of the posterior shape based on the variational free energy and MCMC.

the deviation of the system noise is worse. Deviations may be explained by the fact that the number of observations is relatively small. Furthermore, we have chosen a sample path which contains a transition between the two wells within a small time interval and is thus highly untypical with respect to the prior distribution. This fact was experimentally assessed by estimating the parameters on a sample path without transition, in a time window of the same size. In this case, we obtained estimate roughly within $5\%$ of the true parameter values: $\hat{\sigma} = 0.46$ and $\hat{\theta}_f = 0.92$. Finally, it turns out that our estimate for $\hat{\sigma}$ is close to the one obtained from the MCMC approach as discussed next.

**Posterior distribution over the parameters**

Interestingly, minimizing the free energy $\mathcal{F}_{\sigma^2}$ for different values of $\sigma$ provides us with much more information than a single point estimate for the parameters [14]. Using a suitable prior over $p(\sigma)$, we can approximate the posterior over the system noise variance via

$$p(\sigma^2|Y) \propto e^{-\mathcal{F}_{\sigma^2}} p(\sigma^2), \tag{22}$$

where we take $e^{-\mathcal{F}_{\sigma^2}}$ (at its minimum) as an approximation to the marginal likelihood of the observations $p(Y|\sigma^2)$. To illustrate this point, we assume a non-informative Gamma prior $p(\sigma^2) = \mathcal{G}(\alpha, \beta)$, with $\alpha = 10^{-3}$ and $\beta = 10^{-3}$. A comparison with preliminary MCMC estimates for $p(Y|\sigma^2)$ for $\theta = 1$ and a set of system noise variances indicates that the shape of our approximation is a reasonable indicator of the shape of the posterior. Figure 1(b) shows that at least the mean and the variance of the density come out fairly well.

## 6 Conclusion

We have presented a variational approach to the approximate inference of stochastic differential equations from a finite set of noisy observations. So far, we have tested the method on a one dimensional bi-stable system only. Comparison with a Monte Carlo approach suggests that our method can reproduce the posterior mean fairly well but underestimates the variance in the region of the transition. Parameter estimates also agree well with the MC predictions.

In the future, we will extend our method in various directions. Although our approach is based on a Gaussian approximation of the posterior process, we expect that one can improve on it and obtain non-Gaussian predictions at least for various marginal posterior distributions, including that of the latent variable $X_t$ at a fixed time $t$. This should be possible by generalising our method for the computation of a non-Gaussian shaped probability density for the system noise parameter using the free energy. An important extension of our method will be to systems with many degrees of freedom.

We hope that the possibility of using simpler suboptimal parametrisations of the approximating Gaussian process will allow us to obtain a tractable inference method that scales well to higher dimensions.

**Acknowledgments**

This work has been funded by the EPSRC as part of the Variational Inference for Stochastic Dynamic Environmental Models (VISDEM) project (EP/C005848/1).

## A The Kullback-Leibler divergence interpreted as a path integral

In this section, we show that the Kullback-Leibler divergence between the posterior process $p(X_t|Y, \boldsymbol{\theta}, \boldsymbol{\Sigma})$ and its approximation $q(X|\boldsymbol{\Sigma})$ can be interpreted as a path integral over time. It is an average over all possible realisations, called *sample paths*, of the continuous-time (i.e., infinite dimensional) random variable described by the SDE in the time interval under consideration.

Consider the Euler-Muryama discrete approximation (see for example [13]) of the SDE (1) and its linear approximation (4):

$$\Delta \mathbf{x}_k = \mathbf{f}_k \Delta t + \boldsymbol{\Sigma}^{1/2} \Delta \mathbf{w}_k, \tag{23}$$

$$\Delta \mathbf{x}_k = \mathbf{g}_k \Delta t + \boldsymbol{\Sigma}^{1/2} \Delta \mathbf{w}_k, \tag{24}$$

where $\Delta \mathbf{x}_k \equiv \mathbf{x}_{k+1} - \mathbf{x}_k$ and $\mathbf{w}_k \sim \mathcal{N}(\mathbf{0}, \Delta t \mathbf{I})$. The vectors $\mathbf{f}_k$ and $\mathbf{g}_k$ are shorthand notations for $\mathbf{f}_{\boldsymbol{\theta}}(t_k, \mathbf{x}_k)$ and $\mathbf{g}(t_k, \mathbf{x}_k)$. Hence, the joint distributions of discrete sample paths $\{\mathbf{x}_k\}_{k \geq 0}$ for the true process and its approximation follow from the Markov property:

$$p(\mathbf{x}_0, \dots, \mathbf{x}_K|\boldsymbol{\Sigma}) = p(\mathbf{x}_0) \prod_{k>0} \mathcal{N}(\mathbf{x}_{k+1}|\mathbf{x}_k + \mathbf{f}_k \Delta t, \boldsymbol{\Sigma}\Delta t), \tag{25}$$

$$q(\mathbf{x}_0, \dots, \mathbf{x}_K|\boldsymbol{\Sigma}) = q(\mathbf{x}_0) \prod_{k>0} \mathcal{N}(\mathbf{x}_{k+1}|\mathbf{x}_k + \mathbf{g}_k \Delta t, \boldsymbol{\Sigma}\Delta t), \tag{26}$$

where $p(\mathbf{x}_0)$ is the prior on the intial state $\mathbf{x}_0$ and $q(\mathbf{x}_0)$ is assumed to be Gaussian. Note thate we do not restrict the variational posterior to factorise over the latent states.

The Kullback-Leibler divergence between the two discretized prior processes is given by

$$\mathrm{KL}\left[q\|p\right] = \mathrm{KL}\left[q(\mathbf{x}_0)\|p(\mathbf{x}_0)\right] - \sum_{k>0} \int q(\mathbf{x}_k) \left\langle \ln p(\mathbf{x}_{k+1}|\mathbf{x}_k)\right\rangle_{q(\mathbf{x}_{k+1}|\mathbf{x}_k)} d\mathbf{x}_k$$

$$= \mathrm{KL}\left[q(\mathbf{x}_0)\|p(\mathbf{x}_0)\right] + \frac{1}{2} \sum_{k>0} \left\langle (\mathbf{f}_k - \mathbf{g}_k)^\top \boldsymbol{\Sigma}^{-1} (\mathbf{f}_k - \mathbf{g}_k)\right\rangle_{q(\mathbf{x}_k)} \Delta t,$$

where we omitted the conditional dependency on $\boldsymbol{\Sigma}$ for simplicity. The second term on the right hand side is a sum in $\Delta t$. As a result, taking limits for $\Delta t \to 0$ leads to a proper Riemann integral, which defines an integral over the average sample path:

$$\mathrm{KL}\left[q(X|\boldsymbol{\Sigma})\|p(X|\boldsymbol{\theta}, \boldsymbol{\Sigma})\right] = \mathrm{KL}\left[q_0\|p_0\right] + \frac{1}{2} \int_{t_0}^{t_f} \left\langle (\mathbf{f}_t - \mathbf{g}_t)^\top \boldsymbol{\Sigma}^{-1} (\mathbf{f}_t - \mathbf{g}_t)\right\rangle_{q_t} dt, \tag{27}$$

where $X = \{X_t, \ t_0 \leq t \leq t_f\}$ denotes the stochastic process in the interval $[t_0, t_f]$. The distribution $q_t = q(X_t|\boldsymbol{\Sigma})$ is the marginal at time $t$ for a given system noise covariance $\boldsymbol{\Sigma}$.

It is important to realise that the KL between the induced prior process and its approximation is finite because the system noise covariances are chosen to be identical. If this was not the case, the normalizing constants of $p(\mathbf{x}_{k+1}|\mathbf{x}_k)$ and $q(\mathbf{x}_{k+1}|\mathbf{x}_k)$ would not cancel. This would result in $\mathrm{KL} \to \infty$ when $\Delta t \to 0$.

If we assume that the observations are i.i.d., it follows also that

$$\mathcal{F}_{\boldsymbol{\Sigma}}(q, \boldsymbol{\theta}) = -\sum_n \left\langle \ln p(\mathbf{y}_n|\mathbf{x}_n)\right\rangle_{q(\mathbf{x}_n)} + \mathrm{KL}\left[q(X|\boldsymbol{\Sigma})\|p(X|\boldsymbol{\theta}, \boldsymbol{\Sigma})\right].$$

Clearly, minimising this expression with respect to the variational parameters for a given system noise $\boldsymbol{\Sigma}$ and for a fixed parameter vector $\boldsymbol{\theta}$ is equivalent to minimising the KL between the variational posterior $q(X|\boldsymbol{\Sigma})$ and the true posterior $p(X|Y, \boldsymbol{\theta}, \boldsymbol{\Sigma})$, since the normalizing constant is independent of sample paths.

## B The gradient functions

The general expressions for the gradients of $E_{sde}(t)$ with respect to the variational functions are given by

$$\nabla_{\mathbf{A}} E_{sde}(t) = \mathbf{\Sigma}^{-1} \left\{ \langle \nabla_{\mathbf{x}} \mathbf{f}_{\boldsymbol{\theta}}(t, X_t) \rangle_{q_t} + \mathbf{A}(t) \right\} \mathbf{S}(t) - \nabla_{\mathbf{b}} E_{sde}(t) \mathbf{m}^{\top}(t), \tag{28}$$

$$\nabla_{\mathbf{b}} E_{sde}(t) = \mathbf{\Sigma}^{-1} \left\{ - \langle \mathbf{f}_{\boldsymbol{\theta}}(t, X_t) \rangle_{q_t} - \mathbf{A}(t) \mathbf{m}(t) + \mathbf{b}(t) \right\}, \tag{29}$$

where $\langle \nabla_{\mathbf{x}} \mathbf{f}_{\boldsymbol{\theta}}(t, X_t) \rangle_{q_t} \mathbf{S}(t) = \left\langle \mathbf{f}_{\boldsymbol{\theta}}(t, X_t) \left( X_t - \mathbf{m}(t) \right)^{\top} \right\rangle_{q_t}$ is invoked in order to obtain (28).

## Footnotes

[1]A sample path is a continuous-time realisation of a stochatic process in a certain time interval. Hence, a sample path is an infinite dimensional object.

## References

[1] F. J. Alexander, G. L. Eyink, and J. M. Restrepo. Accelerated Monte Carlo for optimal estimation of time series. *Journal of Statistical Physics*, 119:1331–1345, 2005.

[2] J. D. Annan, J. C. Hargreaves, N. R. Edwards, and R. Marsh. Parameter estimation in an intermediate complexity earth system model using an ensemble Kalman filter. *Ocean Modelling*, 8:135–154, 2005.

[3] A. Apte, M. Hairer, A. Stuart, and J. Voss. Sampling the posterior: An approach to non-Gaussian data assimilation. *Physica D*, 230:50–64, 2007.

[4] C. Archambeau, D. Cornford, M. Opper, and J. Shawe-Taylor. Gaussian process approximation of stochastic differential equations. *Journal of Machine Learning Research: Workshop and Conference Proceedings*, 1:1–16, 2007.

[5] D. Barber. Expectation correction for smoothed inference in switching linear dynamical systems. *Journal of Machine Learning Research*, 7:2515–2540, 2006.

[6] A. Beskos, O. Papaspiliopoulos, G. Roberts, and P. Fearnhead. Exact and computationally efficient likelihood-based estimation for discretely observed diffusion processes (with discussion). *Journal of the Royal Statistical Society B*, 68(3):333–382, 2006.

[7] Christopher M. Bishop. *Pattern Recognition and Machine Learning*. Springer, New York, 2006.

[8] D. Crisan and T. Lyons. A particle approximation of the solution of the Kushner-Stratonovitch equation. *Probability Theory and Related Fields*, 115(4):549–578, 1999.

[9] A. P. Dempster, N. M. Laird, and D. B. Rubin. Maximum likelihood from incomplete data via EM algorithm. *Journal of the Royal Statistical Society B*, 39(1):1–38, 1977.

[10] G. L. Eyink, J. L. Restrepo, and F. J. Alexander. A mean field approximation in data assimilation for nonlinear dynamics. *Physica D*, 194:347–368, 2004.

[11] A. Golightly and D. J. Wilkinson. Bayesian inference for nonlinear multivariate diffusion models observed with error. *Computational Statistics and Data Analysis*, 2007. Accepted.

[12] A. H. Jazwinski. *Stochastic Processes and Filtering Theory*. Academic Press, New York, 1970.

[13] Peter E. Kloeden and Eckhard Platen. *Numerical Solution of Stochastic Differential Equations*. Springer, Berlin, 1999.

[14] H. Lappalainen and J. W. Miskin. Ensemble learning. In M. Girolami, editor, *Advances in Independent Component Analysis*, pages 76–92. Springer-Verlag, 2000.

[15] R. N. Miller, M. Ghil, and F. Gauthiez. Advanced data assimilation in strongly nonlinear dynamical systems. *Journal of the Atmospheric Sciences*, 51:1037–1056, 1994.

[16] Jorge Nocedal and Stephen J. Wright. *Numerical Optimization*. Springer, 2000.

[17] G. Roberts and O. Stramer. On inference for partially observed non-linear diffusion models using the Metropolis-Hastings algorithm. *Biometirka*, 88:603–621, 2001.

